# Principles of real-time computing with feedback applied to cortical microcircuit models

**Wolfgang Maass, Prashant Joshi**
Institute for Theoretical Computer Science
Technische Universitaet Graz
A-8010 Graz, Austria
`maass,joshi@igi.tugraz.at`

**Eduardo D. Sontag**
Department of Mathematics
Rutgers, The State University of New Jersey
Piscataway, NJ 08854-8019, USA
`sontag@cs.rutgers.edu`

## Abstract

The network topology of neurons in the brain exhibits an abundance of feedback connections, but the computational function of these feedback connections is largely unknown. We present a computational theory that characterizes the gain in computational power achieved through feedback in dynamical systems with fading memory. It implies that many such systems acquire through feedback universal computational capabilities for analog computing with a non-fading memory. In particular, we show that feedback enables such systems to process time-varying input streams in diverse ways according to rules that are implemented through internal states of the dynamical system. In contrast to previous attractor-based computational models for neural networks, these flexible internal states are *high-dimensional* attractors of the circuit dynamics, that still allow the circuit state to absorb new information from online input streams. In this way one arrives at novel models for working memory, integration of evidence, and reward expectation in cortical circuits. We show that they are applicable to circuits of conductance-based Hodgkin-Huxley (HH) neurons with high levels of noise that reflect experimental data on in-vivo conditions.

## 1 Introduction

Quite demanding real-time computations with fading memory[1] can be carried out by generic cortical microcircuit models [1]. But many types of computations in the brain, for

example computations that involve memory or persistent internal states, cannot be modeled by such fading memory systems. On the other hand concrete examples of artificial neural networks [2] and cortical microcircuit models [3] suggest that their computational power can be enlarged through feedback from trained readouts. Furthermore the brain is known to have an abundance of feedback connections on several levels: within cortical areas, where pyramidal cells typically have in addition to their long projecting axon a number of local axon collaterals, between cortical areas, and between cortex and subcortical structures. But the computational role of these feedback connections has remained open. We present here a computational theory which characterizes the gain in computational power that a fading memory system can acquire through feedback from trained readouts, both in the idealized case without noise and in the case with noise. This theory simultaneously characterizes the potential gain in computational power resulting from training a few neurons *within* a generic recurrent circuit for a specific task. Applications of this theory to cortical micro-circuit models provide a new way of explaining the possibility of real-time processing of afferent input streams in the light of learning-induced internal circuit states that might represent for example working memory or rules for the timing of behavior. Further details to these results can be found in [4].

## 2 Computational Theory

Recurrent circuits of neurons are from a mathematical perspective special cases of dynamical systems. The subsequent mathematical results show that a large variety of dynamical systems, in particular also neural circuits, can overcome in the presence of feedback the computational limitations of a fading memory – without necessarily falling into the chaotic regime. In fact, feedback endows them with *universal* capabilities for *analog computing*, in a sense that can be made precise in the following way (see Fig. 1A-C for an illustration):

**Theorem 2.1** *A large class $\mathcal{S}_n$ of systems of differential equations of the form*

$$x_i'(t) = f_i(x_1(t), \ldots, x_n(t)) + g_i(x_1(t), \ldots, x_n(t)) \cdot v(t), \quad i = 1, \ldots, n \tag{1}$$

*are in the following sense universal for analog computing:*

*It can respond to an external input $u(t)$ with the dynamics of any $n^{th}$ order differential equation of the form*

$$z^{(n)}(t) = G(z(t), z'(t), z''(t), \ldots, z^{(n-1)}(t)) + u(t) \tag{2}$$

*(for arbitrary smooth functions $G : \mathbb{R}^n \to \mathbb{R}$) if the input term $v(t)$ is replaced by a suitable memoryless feedback function $K(x_1(t), \ldots, x_n(t), u(t))$, and if a suitable memoryless readout function $h(x_1(t), \ldots, x_n(t))$ is applied to its internal state $\langle x_1(t), \ldots, x_n(t) \rangle$.*

*Also the dynamic responses of all systems consisting of several higher order differential equations of the form (2) can be simulated by fixed systems of the form (1) with a corresponding number of feedbacks.*

The class $\mathcal{S}_n$ of dynamical systems that become through feedback universal for analog computing subsumes[2] systems of the form

$$x_i'(t) = -\lambda_i x_i(t) + \sigma \left( \sum_{j=1}^{n} a_{ij} \cdot x_j(t) \right) + b_i \cdot v(t), \quad i = 1, \ldots, n \tag{3}$$

---

$\|\mathbf{u}(\tau) - \tilde{\mathbf{u}}(\tau)\| < \delta$ for all $\tau \in [t - T, t]$. This is a characteristic property of all filters that can be approximated by an integral over the input stream $\mathbf{u}$, or more generally by Volterra- or Wiener series.

[2]for example if the $\lambda_i$ are pairwise different and $a_{ij} = 0$ for all $i, j$, and all $b_i$ are nonzero; fewer restrictions are needed if more then one feedback to the system (3) can be used

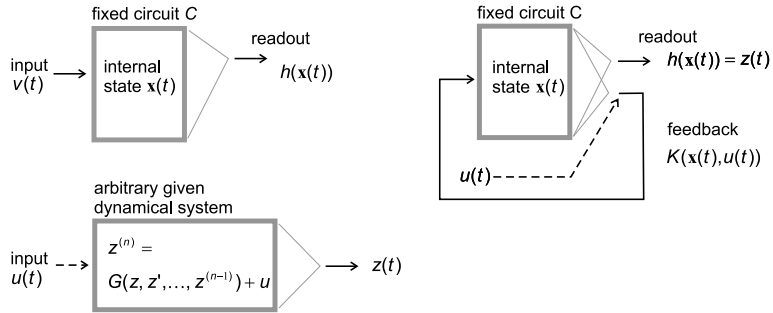

Figure 1: Universal computational capability acquired through feedback according to Theorem 2.1. **(A)** A fixed circuit $C$ with dynamics (1). **(B)** An arbitrary given $n^{th}$ order dynamical system (2) with external input $u(t)$. **(C)** If the input $v(t)$ to circuit $C$ is replaced by a suitable feedback $K(\mathbf{x}(t), u(t))$, then this fixed circuit $C$ can simulate the dynamic response $z(t)$ of the arbitrarily given system shown in B, for any input stream $u(t)$.

that are commonly used to model the temporal evolution of firing rates in neural circuits ($\sigma$ is some standard activation function). If the activation function $\sigma$ is also applied to the term $v(t)$ in (3), the system (3) can still simulate arbitrary differential equations (2) with bounded inputs $u(t)$ and bounded responses $z(t), \ldots, z^{(n-1)}(t)$.

Note that according to [5] all Turing machines can be simulated by systems of differential equations of the form (2). Hence the systems (1) become through feedback also universal for digital computing. A proof of Theorem 2.1 is given in [4].

It has been shown that additive noise, even with an arbitrarily small bounded amplitude, reduces the non-fading memory capacity of any recurrent neural network to some finite number of bits [6, 7]. Hence such network can no longer simulate arbitrary Turing machines. But feedback can still endow noisy fading memory systems with the maximum possible computational power within this a-priori limitation. The following result shows that in principle any finite state machine (= deterministic finite automaton), in particular any Turing machine with tapes of some arbitrary but fixed finite length, can be emulated by a fading memory system with feedback, in spite of noise in the system.

**Theorem 2.2** *Feedback allows linear and nonlinear fading memory systems, even in the presence of additive noise with bounded amplitude, to employ the computational capability and non-fading states of any given finite state machine (in addition to their fading memory) for real-time processing of time varying inputs.*

The precise formalization and the proof of this result (see [4]) are technically rather involved, and cannot be given in this abstract. A key method of the proof, which makes sure that noise does not get amplified through feedback, is also applied in the subsequent computer simulations of cortical microcircuit models. There the readout functions $K$ that provide feedback values $K(\mathbf{x}(t))$ are trained to assume values which cancel the impact of errors or imprecision in the values $K(\mathbf{x}(s))$ of this feedback for immediately preceding time steps $s < t$.

## 3   Application to Generic Circuits of Noisy Neurons

We tested this computational theory on circuits consisting of 600 integrate-and-fire (I&F) neurons and circuits consisting of 600 conductance-based HH neurons, in either case with

a rather high level of noise that reflects experimental data on in-vivo conditions [8]. In addition we used models for dynamic synapses whose individual mixture of paired-pulse depression and facilitation is based on experimental data [9, 10]. Sparse connectivity between neurons with a biologically realistic bias towards short connections was generated by a probabilistic rule, and synaptic parameters were randomly chosen, depending on the type of pre-and postsynaptic neurons, in accordance with these empirical data (see [1] or [4] for details). External inputs and feedback from readouts were connected to populations of neurons within the circuit, with randomly varying connection strengths. The current circuit state $\mathbf{x}(t)$ was modeled by low-pass filtered spike trains from all neurons in the circuit (with a time constant of 30 ms, modeling time constants of receptors and membrane of potential readout neurons). Readout functions $K(\mathbf{x}(t))$ were modeled by weighted sums $\mathbf{w} \cdot \mathbf{x}(t)$

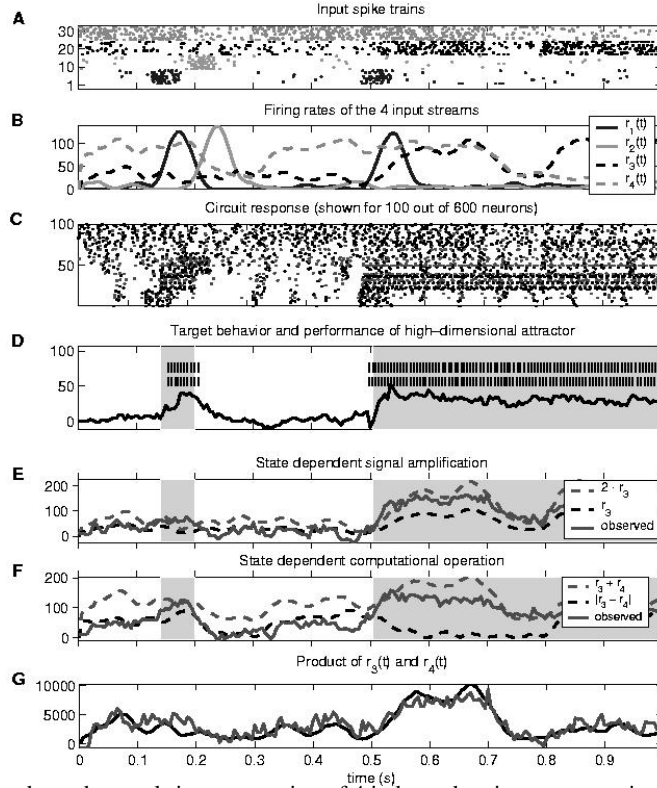

Figure 2: State-dependent real-time processing of 4 independent input streams in a generic cortical microcircuit model. **(A)** 4 input streams, consisting each of 8 spike trains generated by Poisson processes with randomly varying rates $r_i(t), i = 1, \ldots, 4$ (rates plotted in **(B)**; all rates are given in Hz). The 4 input streams and the feedback were injected into disjoint but densely interconnected subpopulations of neurons in the circuit. **(C)** Resulting firing activity of 100 out of the 600 I&F neurons in the circuit. Spikes from inhibitory neurons marked in gray. **(D)** Target activation times of the high-dimensional attractor (gray shading), spike trains of 2 of the 8 I&F neurons that were trained to create the high-dimensional attractor by sending their output spike trains back into the circuit, and average firing rate of all 8 neurons (lower trace). **(E and F)** Performance of linear readouts that were trained to switch their real-time computation task depending on the current state of the high-dimensional attractor: output $2 \cdot r_3(t)$ instead of $r_3(t)$ if the high-dimensional attractor is on (E), output $r_3(t) + r_4(t)$ instead of $|r_3(t) - r_4(t)|$ if the high-dimensional attractor is on (F). **(G)** Performance of linear readout that was trained to output $r_3(t) \cdot r_4(t)$, showing that another linear readout from the same circuit can simultaneously carry out nonlinear computations that are invariant to the current state of the high-dimensional attractor.

whose weights $\mathbf{w}$ were trained during 200 s of simulated biological time to minimize the mean squared error with regard to desired target output functions $K$. After training these weights $\mathbf{w}$ were fixed, and the performance of the otherwise generic circuit was evaluated for new input streams $\mathbf{u}$ (with new input rates drawn from the same distribution) that had not been used for training. It was sufficient to use just linear functions $K$ that transformed the current circuit state $\mathbf{x}(t)$ into a feedback $K(\mathbf{x}(t))$, confirming the predictions of [1] and [2] that the recurrent circuit automatically assumes the role of a kernel (in the sense of machine learning) that creates nonlinear combinations of recent inputs.

We found that computer simulations of such generic cortical microcircuit models confirm the theoretical prediction that feedback from suitably trained readouts enables complex state-dependent real-time processing of a fairly large number of diverse input spike trains within a single circuit (all results shown are for test inputs that had not been used for training). Readout neurons could be trained to turn a high-dimensional attractor on or off in response to particular signals in 2 of the 4 independent input streams (Fig. 2D). The target value for $K(\mathbf{x}(t))$ during training was the currently desired activity-state of the high-dimensional attractor, where $\mathbf{x}(t)$ resulted from giving already tentative spike trains that matched this target value as feedback into the circuit. These neurons were trained to represent in their firing activity at any time the information in which of input streams 1 or 2 a burst had most recently occurred. If it occurred most recently in stream 1, they were trained to fire at 40 Hz, and not to fire otherwise. Thus these neurons were required to represent the non-fading state of a very simple finite state machine, demonstrating in a simple example the validity of Theorem 2.2.

The weights $\mathbf{w}$ of these readout neurons were determined by a sign-constrained linear regression, so that weights from excitatory (inhibitory) presynaptic neurons were automatically positive (negative). Since these readout neurons had the same properties as neurons within the circuit, this computer simulation also provided a first indication of the gain in real-time processing capability that can be achieved by suitable training of a few spiking neurons *within* an otherwise randomly connected recurrent circuit. Fig. 2 shows that other readouts from the same circuit (that do not provide feedback) can be trained to amplify their response to one of the input streams (Fig. 2E), or even switch their computational function (Fig. 2F) if the high-dimensional attractor is in the on-state, thereby providing a model for the way in which internal circuit states can change the "program" for its online processing.

*Continuous* high-dimensional attractors that hold a time-varying analog value (instead of a discrete state) through globally distributed activity within the circuit can be created in the same way through feedback. In fact, several such high-dimensional attractors can co-exist within the same circuit, see Fig. 3B,C,D. This gives rise to a model (Fig. 3) that could explain how timing of behavior and reward expectation are learnt and controlled by neural microcircuits on a behaviorally relevant large time scale. In addition Fig. 4 shows that a continuous high-dimensional attractor that is created through feedback provides a new model for a neural integrator, and that the current value of this neural integrator can be combined within the same circuit and in real-time with variables extracted from time-varying analog input streams.

This learning-induced generation of high-dimensional attractors through feedback provides a new model for the emergence of persistent firing in cortical circuits that does not rely on especially constructed circuits, neurons, or synapses, and which is consistent with high noise (see Fig. 4G for the quite realistic trial-to-trial variability in this circuit of HH neurons with background noise according to [8]). This learning based model is also consistent with the surprising plasticity that has recently been observed even in quite specialized neural integrators [11]. Its robustness can be traced back to the fact that readouts can be trained to correct errors in their previous feedback. Furthermore such error correction is not restricted to linear computational operations, since the inherent kernel property of generic recurrent circuits allows even linear readouts to carry out nonlinear computations on firing rates

(Fig. 2G). Whereas previous models for discrete or continuous attractors in recurrent neural circuits required that the whole dynamics of such circuit was entrained by the attractor, our new model predicts that persistent firing states can co-exist with other high-dimensional attractors and with responses to time-varying afferent inputs within the same circuit. Note that such attractors can equivalently be generated by training (instead of readouts) a few neurons *within* an otherwise generic cortical microcircuit model.

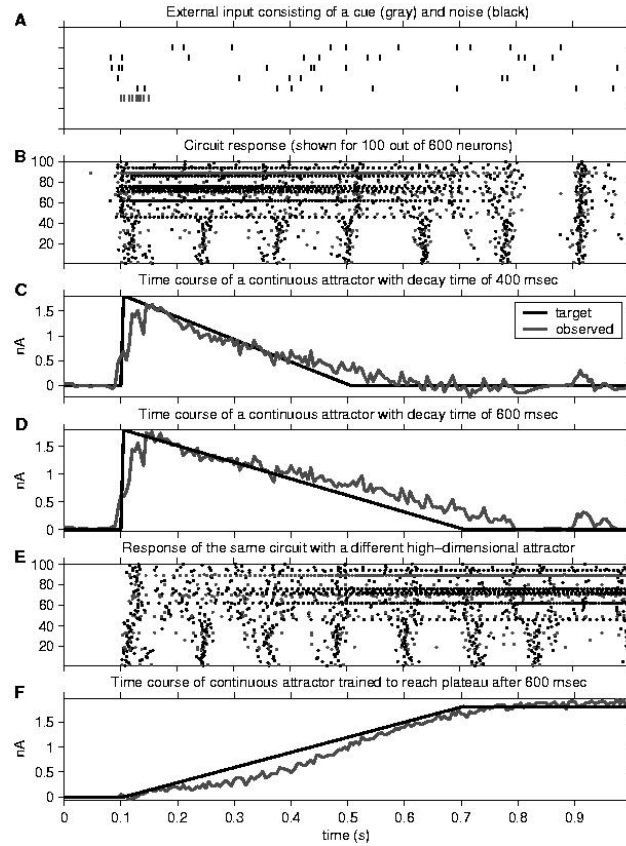

Figure 3: Representation of time for behaviorally relevant time spans in a generic cortical microcircuit model. **(A)** Afferent circuit input, consisting of a cue in one channel (gray) and random spikes (freshly drawn for each trial) in the other channels. **(B)** Response of 100 neurons from the same circuit as in Fig. 2, which has here two co-existing high-dimensional attractors. The autonomously generated periodic bursts with a periodic frequency of about 8 Hz are not related to the task, and readouts were trained to become invariant to them. **(C and D)** Feedback from two linear readouts that were simultaneously trained to create and control two high-dimensional attractors. One of them was trained to decay in 400 ms (C), and the other in 600 ms (D) (scale in nA is the average current injected by feedback into a randomly chosen subset of neurons in the circuit). **(E)** Response of the same neurons as in (B), for the same circuit input, but with feedback from a different linear readout that was trained to create a high-dimensional attractor that increases its activity and reaches a plateau 600 ms after the occurrence of the cue in the input stream. **(F)** Feedback from the linear readout that creates this continuous high-dimensional attractor.

## 4   Discussion

We have demonstrated that persistent memory and online switching of real-time processing can be implemented in generic cortical microcircuit models by training a few neurons

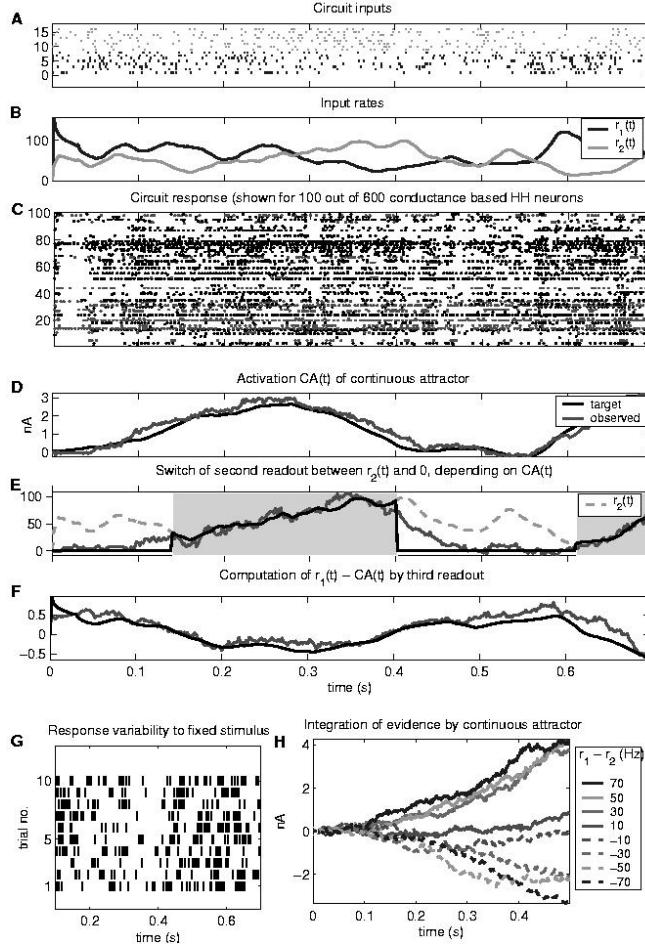

Figure 4: A model for analog real-time computation on external and internal variables in a generic cortical microcircuit (consisting of 600 conductance-based HH neurons). **(A and B)** Two input streams as in Fig. 2; their firing rates $r_1(t), r_2(t)$ are shown in (B). **(C)** Resulting firing activity of 100 neurons in the circuit. **(D)** Performance of a neural integrator, generated by feedback from a linear readout that was trained to output at any time $t$ an approximation $CA(t)$ of the integral $\int_0^t (r_1(s) - r_2(s))ds$ over the difference of both input rates. Feedback values were injected as input currents into a randomly chosen subset of neurons in the circuit. Scale in nA shows average strength of feedback currents (also in panel H). **(E)** Performance of linear readout that was trained to output 0 as long as $CA(t)$ stayed below 1.35 nA, and to output then $r_2(t)$ until the value of $CA(t)$ dropped below 0.45 nA (i.e., in this test run during the shaded time periods). **(F)** Performance of linear readout trained to output $r_1(t) - CA(t)$, i.e. a combination of external and internal variables, at any time $t$ (both $r_1$ and $CA$ normalized into the range $[0, 1]$). **(G)** Response of a randomly chosen neuron in the circuit for 10 repetitions of the same experiment (with input spike trains generated by Poisson processes with the same time-course of firing rates), showing biologically realistic trial-to-trial variability. **(H)** Activity traces of a continuous attractor as in (D), but in 8 different trials for 8 different fixed values of $r_1$ and $r_2$ (shown on the right). The resulting traces are very similar to the temporal evolution of firing rates of neurons in area LIP that integrate sensory evidence (see Fig.5A in [12]).

(within or outside of the circuit) through very simple learning processes (linear regression, or alternatively – with some loss in performance – perceptron learning). The resulting high-dimensional attractors can be made noise-robust through training, thereby overcoming the inherent brittleness of constructed attractors. The high dimensionality of these attractors,

which is caused by the small number of synaptic weights that are fixed for their creation, allows the circuit state to move in or out of other attractors, and to absorb new information from online inputs, while staying within such high-dimensional attractor. The resulting virtually unlimited computational capability of fading memory circuits with feedback can be explained on the basis of the theoretical results that were presented in section 2.

## Acknowledgments

Helpful comments from Wulfram Gerstner, Stefan Haeusler, Herbert Jaeger, Konrad Koerding, Henry Markram, Gordon Pipa, Misha Tsodyks, and Tony Zador are gratefully acknowledged. Written under partial support by the Austrian Science Fund FWF, project # S9102-N04, project # IST2002-506778 (PASCAL) and project # FP6-015879 (FACETS) of the European Union.

## Footnotes

[1]A map (or filter) $F$ from input- to output streams is defined to have fading memory if its current output at time $t$ depends (up to some precision $\varepsilon$) only on values of the input $\mathbf{u}$ during some finite time interval $[t - T, t]$. In formulas: $F$ has fading memory if there exists for every $\varepsilon > 0$ some $\delta > 0$ and $T > 0$ so that $|(F\mathbf{u})(t) - (F\tilde{\mathbf{u}})(t)| < \varepsilon$ for any $t \in \mathbb{R}$ and any input functions $\mathbf{u}, \tilde{\mathbf{u}}$ with

## References

[1] W. Maass, T. Natschläger, and H. Markram. Real-time computing without stable states: A new framework for neural computation based on perturbations. *Neural Computation*, 14(11):2531–2560, 2002.

[2] H. Jäger and H. Haas. Harnessing nonlinearity: predicting chaotic systems and saving energy in wireless communication. *Science*, 304:78–80, 2004.

[3] P. Joshi and W. Maass. Movement generation with circuits of spiking neurons. *Neural Computation*, 17(8):1715–1738, 2005.

[4] W. Maass, P. Joshi, and E. D. Sontag. Computational aspects of feedback in neural circuits. *submitted for publication*, 2005. Online available as #168 from http://www.igi.tugraz.at/maass/.

[5] M. S. Branicky. Universal computation and other capabilities of hybrid and continuous dynamical systems. *Theoretical Computer Science*, 138:67–100, 1995.

[6] M. Casey. The dynamics of discrete-time computation with application to recurrent neural networks and finite state machine extraction. *Neural Computation*, 8:1135 – 1178, 1996.

[7] W. Maass and P. Orponen. On the effect of analog noise in discrete-time analog computations. *Neural Computation*, 10:1071–1095, 1998.

[8] A. Destexhe, M. Rudolph, and D. Pare. The high-conductance state of neocortical neurons in vivo. *Nat. Rev. Neurosci.*, 4(9):739–751, 2003.

[9] H. Markram, Y. Wang, and M. Tsodyks. Differential signaling via the same axon of neocortical pyramidal neurons. *PNAS*, 95:5323–5328, 1998.

[10] A. Gupta, Y. Wang, and H. Markram. Organizing principles for a diversity of GABAergic interneurons and synapses in the neocortex. *Science*, 287:273–278, 2000.

[11] G. Major, R. Baker, E. Aksay, B. Mensh, H. S. Seung, and D. W. Tank. Plasticity and tuning by visual feedback of the stability of a neural integrator. *Proc Natl Acad Sci*, 101(20):7739–7744, 2004.

[12] M. E. Mazurek, J. D. Roitman, J. Ditterich, and M. N. Shadlen. A role for neural integrators in perceptual decision making. *Cerebral Cortex*, 13(11):1257–1269, 2003.
